# Bayesian Color Constancy
# with Non-Gaussian Models

**Charles Rosenberg**
Computer Science Department
Carnegie Mellon University
Pittsburgh, PA 15213
chuck@cs.cmu.edu

**Thomas Minka**
Statistics Department
Carnegie Mellon University
Pittsburgh, PA 15213
minka@stat.cmu.edu

**Alok Ladsariya**
Computer Science Department
Carnegie Mellon University
Pittsburgh, PA 15213
alokl@cs.cmu.edu

## Abstract

We present a Bayesian approach to color constancy which utilizes a non-Gaussian probabilistic model of the image formation process. The parameters of this model are estimated directly from an uncalibrated image set and a small number of additional algorithmic parameters are chosen using cross validation. The algorithm is empirically shown to exhibit RMS error lower than other color constancy algorithms based on the Lambertian surface reflectance model when estimating the illuminants of a set of test images. This is demonstrated via a direct performance comparison utilizing a publicly available set of real world test images and code base.

## 1 Introduction

Color correction is an important preprocessing step for robust color-based computer vision algorithms. Because the illuminants in the world have varying colors, the measured color of an object will change under different light sources. We propose an algorithm for color constancy which, given an image, will automatically estimate the color of the illuminant (assumed constant over the image), allowing the image to be color corrected.

This color constancy problem is ill-posed, because object color and illuminant color are not uniquely separable. Historically, algorithms for color constancy have fallen into two groups. The first group imposes constraints on the scene and/or the illuminant, in order to remove the ambiguities. The second group uses a statistical model to quantify the probability of each illuminant and then makes an estimate from these probabilities. The statistical approach is attractive, since it is more general and more automatic—hard constraints are a special case of statistical models, and they can be learned from data instead of being specified in advance. But as shown by [3, 1], currently the best performance on real images is achieved by gamut mapping, a constraint-based algorithm. And, in the words of some leading researchers, even gamut mapping is not "good enough" for object recognition [8].

In this paper, we show that it is possible to outperform gamut mapping with a statistical approach, by using appropriate probability models with the appropriate statistical framework. We use the principled Bayesian color constancy framework of [4], but combine it with rich, nonparametric image models, such as used by Color by Correlation [1]. The

result is a Bayesian algorithm that works well in practice and addresses many of the issues with Color by Correlation, the leading statistical algorithm [1].

At the same time, we suggest that statistical methods still have much to learn from constraint-based methods. Even though our algorithm outperforms gamut mapping on average, there are cases in which gamut mapping provides better estimates, and, in fact, the errors of the two methods are surprisingly uncorrelated. This is an interesting result, because it suggests that gamut mapping exploits image properties which are different from what is learned by our algorithm, and probably other statistical algorithms. If this is true, and if our statistical model could be extended in a way that captures these additional properties, better algorithms should be possible in the future.

## 2   The imaging model

Our approach is to model the observed image pixels with a probabilistic generative model, decomposing them as the product of unknown surface reflectances with an unknown illuminant. Using Bayes' rule, we obtain a posterior for the illuminant, and from this we extract the estimate with minimum risk, e.g., the minimum expected chromaticity error.

Let $\mathbf{y}$ be an image pixel with three color channels: $(y_r, y_g, y_b)$. The pixel is assumed to be the result of light reflecting off of a surface under the Lambertian reflectance model. Denote the power of the light in each channel by $\boldsymbol{\ell} = (\ell_r, \ell_g, \ell_b)$, with each channel ranging from zero to infinity. For each channel, a surface can reflect none of the light, all of the light, or somewhere in between. Denote this reflectance by $\mathbf{x} = (x_r, x_g, x_b)$, with each channel ranging from zero to one. The model for the pixel is the well-known diagonal lighting model:

$$y_r = \ell_r x_r \qquad y_g = \ell_g x_g \qquad y_b = \ell_b x_b \qquad (1)$$

To simplify the equations below, we write this in matrix form as

$$\mathbf{L} = \mathrm{diag}(\boldsymbol{\ell}) \qquad (2)$$
$$\mathbf{y} = \mathbf{Lx} \qquad (3)$$

This specifies the conditional distribution $p(\mathbf{y}|\boldsymbol{\ell}, \mathbf{x})$. In reality, there are sensor noise and other factors which affect the observed color, but we will consider these to be negligible.

Next we make the common assumption that the light and the surface have been chosen independently, so that $p(\boldsymbol{\ell}, \mathbf{x}) = p(\boldsymbol{\ell})p(\mathbf{x})$. The prior distribution for the illuminant ($p(\boldsymbol{\ell})$) will be uniform over a constraint set, described later in section 5.3.

The most difficult step is to construct a model for the surface reflectances in an image containing many pixels:

$$\mathbf{Y} = (\mathbf{y}(1), ..., \mathbf{y}(n)) \qquad (4)$$
$$\mathbf{X} = (\mathbf{x}(1), ..., \mathbf{x}(n)) \qquad (5)$$

We need a distribution $p(\mathbf{X})$ for all $n$ reflectances. One approach is to assume that the reflectances are independent and Gaussian, as in [4], which gives reasonable results but can be improved upon. Our approach is to quantize the reflectance vectors into $K$ bins, and consider the reflectances to be *exchangeable*—a weaker assumption than independence. Exchangeability implies that the probability only depends on the number of reflectances in each bin. Thus if we denote the *reflectance histogram* by $(n_1, ..., n_K)$, where $\sum_k n_k = n$, then

$$p(\mathbf{x}(1), ..., \mathbf{x}(n)) \propto f(n_1, ..., n_K) \qquad (6)$$

where $f$ is a function to be specified. Independence is a special case of exchangeability. If $m_k$ is the probability of a surface having a reflectance value in bin $k$, so that $\sum_k m_k = 1$,

then independence says

$$f(n_1, ..., n_K) = \prod_k m_k^{n_k} \tag{7}$$

As an alternative to this, we have experimented with the Dirichlet-multinomial model, which employs a parameter $s > 0$ to control the amount of correlation. Under this model,

$$f(n_1, ..., n_K) = \frac{\Gamma(s)}{\Gamma(n+s)} \prod_k \frac{\Gamma(n_k + sm_k)}{\Gamma(sm_k)} \tag{8}$$

For large $s$, correlation is weak and the model reduces to (7). For small $s$, correlation is strong and the model expects a few reflectances to be repeated many times, which is what we see in real images. When $s$ is very small, the expression (8) can be reduced to a simple form:

$$f(n_1, ..., n_K) \approx \frac{1}{s\Gamma(n)} \prod_k (sm_k\Gamma(n_k))^{\mathrm{clip}(n_k)} \tag{9}$$

$$\mathrm{clip}(n_k) = \begin{cases} 0 & \text{if } n_k = 0 \\ 1 & \text{if } n_k > 0 \end{cases} \tag{10}$$

This resembles a multinomial distribution on clipped counts. Unfortunately, this distribution strongly prefers that the image contains a small number of different reflectances, which biases the light source estimate. Empirically we have achieved our best results using a "normalized count" modification of the model which removes this bias:

$$f(n_1, ..., n_K) = \prod_k m_k^{\nu_k} \tag{11}$$

$$\nu_k = n\frac{\mathrm{clip}(n_k)}{\sum_k \mathrm{clip}(n_k)} \tag{12}$$

The modified counts $\nu_k$ sum to $n$ just like the original counts $n_k$, but are distributed equally over all reflectances present in the image.

## 3   The color constancy algorithm

The algorithm for estimating the illuminant has two parts: (1) discretize the set of all illuminants on a fine grid and compute their likelihood and (2) pick the illuminant which minimizes the risk.

The likelihood of the observed image data $\mathbf{Y}$ for a given illuminant $\boldsymbol{\ell}$ is

$$p(\mathbf{Y}|\boldsymbol{\ell}) = \int_\mathbf{X} \left( \prod_i p(\mathbf{y}(i)|\boldsymbol{\ell}, \mathbf{x}(i)) \right) p(\mathbf{X})d\mathbf{X} \tag{13}$$

$$= |\mathbf{L}^{-1}|^n p(\mathbf{X} = \mathbf{L}^{-1}\mathbf{Y}) \tag{14}$$

The quantity $\mathbf{L}^{-1}\mathbf{Y}$ can be understood as the *color-corrected image*. The determinant term, $1/(\ell_r\ell_g\ell_b)^n$, makes this a valid distribution over $\mathbf{Y}$ and has the effect of introducing a preference for dimmer illuminants independently of the prior on reflectances. Also implicit in this likelihood are the bounds on $\mathbf{x}$, which require reflectances to be in the range of zero and one and thus we restrict our search to illuminants that satisfy:

$$\ell_r \geq \max_i y_r(i) \qquad \ell_g \geq \max_i y_g(i) \qquad \ell_b \geq \max_i y_b(i) \tag{15}$$

The posterior probability for $\boldsymbol{\ell}$ then follows:

$$p(\boldsymbol{\ell}|\mathbf{Y}) \propto p(\mathbf{Y}|\boldsymbol{\ell})p(\boldsymbol{\ell}) \tag{16}$$

$$\propto |\mathbf{L}^{-1}|^n p(\mathbf{X} = \mathbf{L}^{-1}\mathbf{Y})p(\boldsymbol{\ell}) \tag{17}$$

The next step is to find the estimate of $\ell$ with minimum risk. An answer that the illuminant is $\ell^*$, when it is really $\ell$, incurs some cost, denoted $R(\ell^*|\ell)$. Let this function be quadratic in some transformation $\mathbf{g}$ of the illuminant vector $\ell$:

$$R(\ell^*|\ell) \;\; = \;\; ||\mathbf{g}(\ell^*) - \mathbf{g}(\ell)||^2 \tag{18}$$

This occurs, for example, when the cost function is squared error in chromaticity. Then the minimum-risk estimate satisfies

$$\mathbf{g}(\ell^*) \;\; = \;\; \int_{\ell} \mathbf{g}(\ell)p(\ell|\mathbf{Y})d\ell \tag{19}$$

The right-hand side, the posterior mean of $\mathbf{g}$, and the normalizing constant of the posterior can be computed in a single loop over the grid of illuminants.

## 4   Relation to other algorithms

In this section we describe related color constancy algorithms using the framework of the imaging model introduced in section 2. This is helpful because it allows us to compare all of these algorithms in a single framework and understand the assumptions made by each.

**Independent, Gaussian reflectances**   The previous work most similar to our own is by [10] and [4]; however, these methods are not tested on real images. They use a similar imaging model and maximum-likelihood and minimum-risk estimation, respectively. The difference is that they use a Gaussian prior for the reflectance vectors, and assume the reflectances for different pixels are independent. The Gaussian assumption leads to a simple likelihood formula whose maximum can be found by gradient methods. However, as mentioned by [4], this is a constraining assumption, and more appropriate priors would be preferable.

**Scale by max**   The scale by max algorithm (as tested e.g. in [3]) estimates the illuminant by the simple formula

$$\ell_r = \max_i y_r(i) \qquad \ell_g = \max_i y_g(i) \qquad \ell_b = \max_i y_b(i) \tag{20}$$

which is the dimmest illuminant in the valid set (15). In the Bayesian algorithm, this solution can be achieved by letting the reflectances be independent and uniform over the range 0 to 1. Then $p(\mathbf{X})$ is constant and the maximum-likelihood illuminant is (20). This connection was also noticed by [4].

**Gray-world**   The gray-world algorithm [5] chooses the illuminant such that the average value in each channel of the corrected image is a constant, e.g. 0.5. This is equivalent to the Bayesian algorithm with a particular reflectance prior. Let the reflectances be independent for each pixel and each channel, with distribution $p(x_c) \propto \exp(-2x_c)$ in each channel $c$. The log-likelihood for $\ell_c$ is then

$$\log p(\mathbf{Y}_c|\ell_c) = -n\log\ell_c - 2\sum_i \frac{y_c(i)}{\ell_c} + \text{const.} \tag{21}$$

whose maximum is (as desired)

$$\ell_c = \frac{2}{n}\sum_i y_c(i) \tag{22}$$

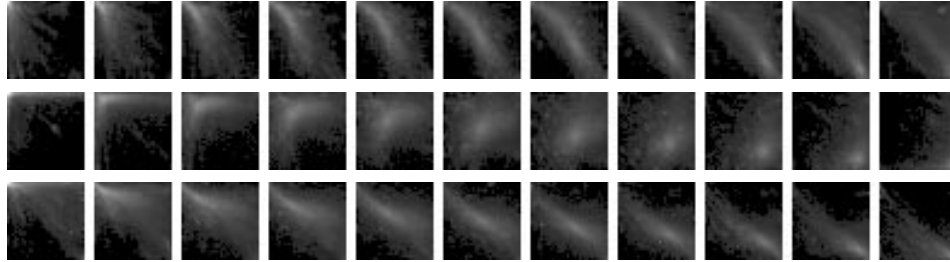

Figure 1: Plots of slices of the three dimensional color surface reflectance distribution along a single dimension. Row one plots green versus blue with 0,0 at the upper left of each subplot and slices in red whose magnitude increases from left to right. Row two plots red versus blue with slices in green. Row three plots red versus green with slices in blue.

**Color by Correlation** Color by Correlation [6, 1] also uses a likelihood approach, but with a different imaging model that is not based on reflectance. Instead, observed pixels are quantized into color bins, and the frequency of each bin is counted for each illuminant, in a finite set of illuminants. (Note that this is different from quantizing *reflectances*, as done in our approach.) Let $m_k(\boldsymbol{\ell})$ be the frequency of color bin $k$ for illuminant $\boldsymbol{\ell}$, and let $n_1 \cdots n_K$ be the color histogram of the image, then the likelihood of $\boldsymbol{\ell}$ is computed as

$$p(\mathbf{Y}|\boldsymbol{\ell}) \quad = \quad \prod_k m_k(\boldsymbol{\ell})^{\mathrm{clip}(n_k)} \tag{23}$$

While theoretically this is very general, there are practical limitations. First there are training issues. One must learn the color frequencies for every possible illuminant. Since collecting real-world data whose illuminant is known is difficult, $m_k(\boldsymbol{\ell})$ is typically trained synthetically with random surfaces, which may not represent the statistics of natural scenes. The second issue is that colors and illuminants live in an unbounded 3D space [1], unlike reflectances which are bounded. In order to store a color distribution for each illuminant, brightness variation needs to be artificially bounded. The third issue is storage. To reduce the storage of the $m_k(\boldsymbol{\ell})$'s, Barnard et al [1] store the color distribution only for illuminants of a fixed brightness. However, as they describe, this introduces a bias in the estimation they refer to as the "discretization problem" and try to solve it by penalizing bright illuminants. The other part of the bias is due to using clipped counts in the likelihood. As explained in section 2, a multinomial likelihood with clipped counts is a special case of the Dirichlet-multinomial, and prefers images with a small number of different colors. This bias can be removed using a different likelihood function, such as (11).

## 5 Parameter estimation

### 5.1 Reflectance Distribution

To implement the Bayesian algorithm, we need to learn the real-world frequencies $m_k$ of quantized reflectance vectors. The direct approach to this would require a set of images with ground truth information regarding the associated illumination parameters or, alternately, a set of images captured under a canonical illuminant and camera.

Unfortunately, it is quite difficult to collect a large number of images under controlled conditions. To avoid this issue, we use *bootstrapping*, as described in [9], to approximate the ground truth. The estimates from some "base" color constancy algorithm are used as a proxy for the ground truth. This might seem to be problematic in that it would limit any algorithm based on these estimates to perform only as well as the base algorithm. However, this need not be the case if the errors made by the base algorithm are relatively unbiased.

We used approximately 2300 randomly selected JPEG images from news sites on the web for bootstrapping, consisting mostly of outdoor scenes, indoor news conferences, and sporting event scenes. The scale by max algorithm was used as our "base" algorithm. Figure 1 is a plot of the probability distribution collected, where lighter regions represent higher probability values. The distribution is highly structured and varies with the magnitude of the channel response. This structure is important because it allows our algorithm to disambiguate between potential solutions to the ill-posed illumination estimation problem.

### 5.2 Pre-processing and quantization

To increase robustness, pre-processing is performed on the image, similar to that performed in [3]. The first pre-processing step scales down the image to reduce noise and speed up the algorithm. A new image is formed in which each pixel is the mean of an $m$ by $m$ block of the original image. The second pre-processing step removes dark pixels from the computation, which, because of noise and quantization effects do not contain reliable color information. Pixels whose $y_r + y_g + y_b$ channel sum is less than a given threshold are excluded from the computation.

In addition to the reflectance prior, the parameters of our algorithm are: the number of reflectance histogram bins, the scale down factor, and the dark pixel threshold value. To set these parameters values, the algorithm was run over a large grid of parameter variations and performance on the tuning set was computed. The tuning set was a subset of the "model" data set described in [7] and disjoint from the test set. A total of 20 images were used, 10 objects imaged under 2 illuminants. (The "ball2" object was removed so that there was no overlap between the tuning and test sets.) For the purpose of speed, only images captured with the Philips Ultralume and the Macbeth Judge II fluorescent illuminants were included.

The best set of parameters was found to be: $32 \times 32 \times 32$ reflectance bins, scale down by $m = 3$, and omit pixels with a channel sum less than $8/(3 \times 255)$.

### 5.3 Illuminant prior

To facilitate a direct comparison, we adopt the two illuminant priors from [3]. Each is uniform over a subset of illuminants. The first prior, *full set*, discretizes the illuminants uniformly in polar coordinates. The second prior, *hull set*, is a subset of *full set* restricted to be within the convex hull of the test set illuminants and other real world illuminants. Overall brightness, $\ell_r + \ell_g + \ell_b$, is discretized in the range of 0 to 6 in 0.01 steps.

## 6 Experiments

### 6.1 Evaluation Specifics

To test the algorithms we use the publicly available real world image data set [2] used by Barnard, Martin, Coath and Funt in a comprehensive evaluation of color constancy algorithms in [3]. The data set consists of images of 30 scenes captured under 11 light sources, for a total of 321 images (after the authors removed images which had collection problems) with ground truth illuminant information provided in the form of an RGB value.

As in the "rg error" measure of [3], illuminant error is measured in chromaticity space:

$$\ell_1 = \ell_r/(\ell_r + \ell_g + \ell_b) \qquad \ell_2 = \ell_g/(\ell_r + \ell_g + \ell_b) \tag{24}$$

$$R(\boldsymbol{\ell}^*|\boldsymbol{\ell}) = (\ell_1^* - \ell_1)^2 + (\ell_2^* - \ell_2)^2 \tag{25}$$

The Bayesian algorithm is adapted to minimize this risk by computing the posterior mean in chromaticity space. The performance of an algorithm on the test set is reported as the square root of the average $R(\boldsymbol{\ell}^*|\boldsymbol{\ell})$ across all images, referred to as the RMS error.

Table 1: The average error of several color constancy algorithms on the test set. The value in parentheses is 1.64 times the standard error of the average, so that if two error intervals do not overlap the difference is significant at the 95% level.

| Algorithm | RMS Error for Full Set | RMS Error for Hull Set |
|---|---|---|
| Scale by Max | 0.0584 (+/- 0.0034) | 0.0584 (+/- 0.0034) |
| Gamut Mapping without Segmentation | 0.0524 (+/- 0.0029) | 0.0461 (+/- 0.0025) |
| Gamut Mapping with Segmentation | 0.0426 (+/- 0.0023) | 0.0393 (+/- 0.0021) |
| Bayes with Bootstrap Set Model | 0.0442 (+/- 0.0025) | 0.0351 (+/- 0.0020) |
| Bayes with Tuning Set Model | 0.0344 (+/- 0.0017) | 0.0317 (+/- 0.0017) |

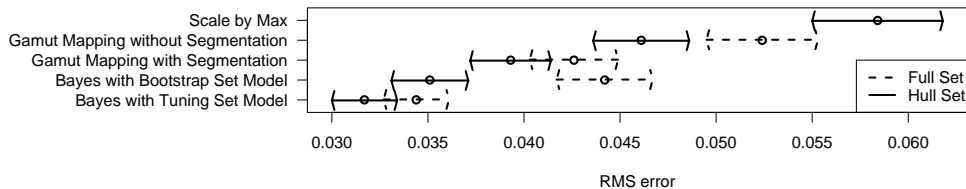

Figure 2: A graphical rendition of table 1. The standard errors are scaled by 1.64, so that if two error bars do not overlap the difference is significant at the 95% level.

## 6.2 Results

The results[1] are summarized in Table 1 and Figure 2. We compare two versions of our Bayesian method to the gamut mapping and scale by max algorithms. The appropriate preprocessing for each algorithm was applied to the images to achieve the best possible performance. (Note that we do not include results for color by correlation since the gamut mapping results were found to be significantly better in [3].) In all configurations, our algorithm exhibits the lowest RMS error except in a single case where it is not statistically different than that of gamut mapping. The differences for the hull set are especially large. The hull set is clearly a useful constraint that improves the performance of all of the algorithms evaluated.

The two versions of our Bayesian algorithm differ only in the data set used to build the reflectance prior. The tuning set, while composed of separate images than the test set, is very similar and has known illuminants, and, accordingly, gives the best results. Yet the performance when trained on a very different set of images, the uncalibrated bootstrap set of section 5.1, is not that different, particularly when the illuminant search is constrained.

The gamut mapping algorithm (called CRULE and ECRULE in [3]) is also presented in two versions: with and without segmenting the images as a preprocessing step as described in [3]. These results were computed using software provided by Barnard and used to generate the results in [3]. In the evaluation of color constancy algorithms in [3] gamut mapping was found on average to outperform all other algorithms when evaluated on real world images.

It is interesting to note that the gamut mapping algorithm is sensitive to segmentation. Since fundamentally it should not be sensitive to the number of pixels of a particular color in the image we must assume that this is because the segmentation is implementing some form of noise filtering. The Bayesian algorithm currently does not use segmentation.

Scale by max is also included as a reference point and still performs quite well given its simplicity, often beating out much more complex constancy algorithms [8, 3]. Its performance is the same for both illuminant sets since it does not involve a search over illuminants.

Surprisingly, when the error of the Bayesian method is compared with the gamut mapping method on individual test images, the correlation coefficient is -0.04. Thus the images which confuse the Bayesian method are quite different from the images which confuse gamut mapping. This suggests that an algorithm which could jointly model the image properties exploited by both algorithms might give dramatic improvements. As an example of the potential improvement, the RMS error of an ideal algorithm whose error is the minimum of Bayes and gamut on each image in the test set is only 0.019.

## 7    Conclusions and Future Work

We have demonstrated empirically that Bayesian color constancy with the appropriate non-Gaussian models can outperform gamut mapping on a standard test set. This is true regardless of whether a calibrated or uncalibrated training set is used, or whether the full set or a restricted set of illuminants is searched. This should give new hope to the pursuit of statistical methods as a unifying framework for color constancy.

The results also suggest ways to improve the Bayesian algorithm. The particular image model we have used, the normalized count model, is only one of many that could be tried. This is simply an image modeling problem which can be attacked using standard statistical methods. A particularly promising direction is to pursue models which can enforce constraints like that in the gamut mapping algorithm, since the images where Bayes has the largest errors appear to be relatively easy for gamut mapping.

**Acknowledgments**

We would like to thank Kobus Barnard for making his test images and code publicly available. We would also like to thank Martial Hebert for his valuable insight and advice and Daniel Huber and Kevin Watkins for their help in revising this document. This work was sponsored in part by a fellowship from the Eastman Kodak company.

**References**

[1]  K. Barnard, L. Martin, and B. Funt, "Colour by correlation in a three dimensional colour space," *Proceedings of the 6th European Conference on Computer Vision*, pp. 275–289, 2000.

[2]  K. Barnard, L. Martin, B. Funt, and A. Coath, "A data set for colour research," *Color Research and Application*, Volume 27, Number 3, pp. 147-151, 2002, http://www.cs.sfu.ca/~colour/data/colour_constancy_test_images/

[3]  K. Barnard, L. Martin, A. Coath, and B. Funt, "A comparison of color constancy algorithms; Part Two. Experiments with Image Data," *IEEE Transactions in Image Processing*, vol. 11. no. 9. pp. 985-996, 2002.

[4]  D. H. Brainard and W. T. Freeman, "Bayesian color constancy," *Journal of the Optical Society of America A*, vol. 14, no. 7, pp. 1393-1411, 1997.

[5]  G. Buchsbaum, "A spatial processor model for object colour perception," *Journal of the Franklin Institute*, vol. 10, pp. 1-26, 1980.

[6]  G. D. Finlayson and S. D. Hordley and P. M. Hubel, "Colour by correlation: a simple, unifying approach to colour constancy," *The Proceedings of the Seventh IEEE International Conference on Computer Vision*, vol. 2, pp. 835-842, 1999.

[7]  B. Funt and V. Cardei and K. Barnard, "Learning color constancy," *Proceedings of Imaging Science and Technology / Society for Information Display Fourth Color Imaging Conference.* pp. 58-60, 1996.

[8]  B. Funt and K. Barnard and L. Martin, "Is colour constancy good enough?," *Proceedings of the Fifth European Conference on Computer Vision,* pp. 445-459, 1998.

[9]  B. Funt and V. Cardei. "Bootstrapping color constancy," *Proceedings of SPIE: Electronic Imaging* IV, 3644, 1999.

[10]  H. J. Trussell and M. J. Vrhel, "Estimation of illumination for color correction," *Proc ICASSP*, pp. 2513-2516, 1991.

## Footnotes

[1]Result images can be found at `http://www.cs.cmu.edu/~chuck/nips-2003/`
